# Probabilistic Multi-Task Feature Selection

**Yu Zhang**[1]**, Dit-Yan Yeung**[1]**, Qian Xu**[2]
[1]Department of Computer Science and Engineering, [2]Bioengineering Program
Hong Kong University of Science and Technology
{zhangyu,dyyeung}@cse.ust.hk, fleurxq@ust.hk

## Abstract

Recently, some variants of the $l_1$ norm, particularly matrix norms such as the $l_{1,2}$ and $l_{1,\infty}$ norms, have been widely used in multi-task learning, compressed sensing and other related areas to enforce sparsity via joint regularization. In this paper, we unify the $l_{1,2}$ and $l_{1,\infty}$ norms by considering a family of $l_{1,q}$ norms for $1 < q \le \infty$ and study the problem of determining the most appropriate sparsity enforcing norm to use in the context of multi-task feature selection. Using the generalized normal distribution, we provide a probabilistic interpretation of the general multi-task feature selection problem using the $l_{1,q}$ norm. Based on this probabilistic interpretation, we develop a probabilistic model using the noninformative Jeffreys prior. We also extend the model to learn and exploit more general types of pairwise relationships between tasks. For both versions of the model, we devise expectation-maximization (EM) algorithms to learn all model parameters, including $q$, automatically. Experiments have been conducted on two cancer classification applications using microarray gene expression data.

## 1 Introduction

Learning algorithms based on $l_1$ regularization have a long history in machine learning and statistics. A well-known property of $l_1$ regularization is its ability to enforce sparsity in the solutions. Recently, some variants of the $l_1$ norm, particularly matrix norms such as the $l_{1,2}$ and $l_{1,\infty}$ norms, were proposed to enforce sparsity via joint regularization [24, 17, 28, 1, 2, 15, 20, 16, 18]. The $l_{1,2}$ norm is the sum of the $l_2$ norms of the rows and the $l_{1,\infty}$ norm is the sum of the $l_\infty$ norms of the rows. Regularizers based on these two matrix norms encourage row sparsity, i.e., they encourage entire rows of the matrix to have zero elements. Moreover, these norms have also been used for enforcing group sparsity among features in conventional classification and regression problems, e.g., group LASSO [29]. Recently, they have been widely used in multi-task learning, compressed sensing and other related areas. However, when given a specific application, we often have no idea which norm is the most appropriate choice to use.

In this paper, we study the problem of determining the most appropriate sparsity enforcing norm to use in the context of multi-task feature selection [17, 15]. Instead of choosing between specific choices such as the $l_{1,2}$ and $l_{1,\infty}$ norms, we consider a family of $l_{1,q}$ norms. We restrict $q$ to the range $1 < q \le \infty$ to ensure that all norms in this family are convex, making it easier to solve the optimization problem formulated based on it. Within this family, the $l_{1,2}$ and $l_{1,\infty}$ norms are just two special cases. Using the $l_{1,q}$ norm, we formulate the general multi-task feature selection problem and give it a probabilistic interpretation. It is noted that the automatic relevance determination (ARD) prior [9, 3, 26] comes as a special case under this interpretation. Based on this probabilistic interpretation, we develop a probabilistic formulation using a noninformative prior called the Jeffreys prior [10]. We devise an expectation-maximization (EM) algorithm [8] to learn all model parameters, including $q$, automatically. Moreover, an underlying assumption of existing multi-task feature selection methods is that all tasks are similar to each other and they share the same features. This assumption may not be correct in practice because there may exist outlier tasks

or tasks with negative correlation. As another contribution of this paper, we propose to use a matrix variate generalized normal prior [13] for the model parameters to learn the relationships between tasks. The task relationships learned here can be seen as an extension of the task covariance used in [4, 32, 31]. Experiments will be reported on two cancer classification applications using microarray gene expression data.

## 2  Multi-Task Feature Selection

Suppose we are given $m$ learning tasks $\{T_i\}_{i=1}^m$. For the $i$th task $T_i$, the training set $\mathcal{D}_i$ consists of $n_i$ labeled data points in the form of ordered pairs $(\mathbf{x}_j^i, y_j^i)$, $j = 1, \ldots, n_i$, with $\mathbf{x}_j^i \in \mathbb{R}^d$ and its corresponding output $y_j^i \in \mathbb{R}$ if it is a regression problem and $y_j^i \in \{-1, 1\}$ if it is a binary classification problem. The linear function for $T_i$ is defined as $f_i(\mathbf{x}) = \mathbf{w}_i^T \mathbf{x} + b_i$. For applications that need feature selection, e.g., document classification, the feature dimensionality is usually very high and it has been found that linear methods usually perform better.

The objective functions of most existing multi-task feature selection methods [24, 17, 28, 1, 2, 15, 20, 16, 18] can be expressed in the following form:

$$\sum_{i=1}^m \sum_{j=1}^{n_i} L(y_j^i, \mathbf{w}_i^T \mathbf{x}_j^i + b_i) + \lambda R(\mathbf{W}), \tag{1}$$

where $\mathbf{W} = (\mathbf{w}_1, \ldots, \mathbf{w}_m)$, $L(\cdot, \cdot)$ denotes the loss function (e.g., squared loss for regression and hinge loss for classification), $R(\cdot)$ is the regularization function that enforces feature sparsity under the multi-task setting, and $\lambda$ is the regularization parameter controlling the relative contribution of the empirical loss and the regularizer. Multi-task feature selection seeks to minimize the objective function above to obtain the optimal parameters $\{\mathbf{w}_i, b_i\}$. Two regularization functions are widely used in existing multi-task feature selection methods. One of them is based on the $l_{1,2}$ norm of $\mathbf{W}$ [17, 28, 1, 2, 16, 18]: $R(\mathbf{W}) = \sum_{k=1}^d \|\mathbf{w}^k\|_2$ where $\|\cdot\|_q$ denotes the $q$-norm (or $l_q$ norm) of a vector and $\mathbf{w}^k$ denotes the $k$th row of $\mathbf{W}$. Another one is based on the $l_{1,\infty}$ norm of $\mathbf{W}$ [24, 15, 20]: $R(\mathbf{W}) = \sum_{k=1}^d \|\mathbf{w}^k\|_\infty$.

In this paper, we unify these two cases by using the $l_{1,q}$ norm of $\mathbf{W}$ to define a more general regularization function:

$$R(\mathbf{W}) = \sum_{k=1}^d \|\mathbf{w}^k\|_q, \qquad 1 < q \le \infty.$$

Note that when $q < 1$, $R(\mathbf{W})$ is non-convex with respect to $\mathbf{W}$. Although $R(\mathbf{W})$ is convex when $q = 1$, each element of $\mathbf{W}$ is independent of each other and so the regularization function cannot enforce feature sparsity. Thus we restrict the range to $1 < q \le \infty$.

Even though restricting the range to $1 < q \le \infty$ can enforce feature sparsity between different tasks, different values of $q$ imply different 'group discounts' for sharing the same feature. Specifically, when $q$ approaches 1, the cost grows almost linearly with the number of tasks that use a feature, and when $q = \infty$, only the most demanding task matters. So selecting a proper $q$ can potentially have a significant effect on the performance of the learning algorithms.

In the following, we first give a probabilistic interpretation for multi-task feature selection methods. Based on this probabilistic interpretation, we then develop a probabilistic model which, among other things, can solve the model selection problem automatically by estimating $q$ from data.

## 3  Probabilistic Interpretation

In this section, we will show that existing multi-task feature selection methods are related to the maximum a posteriori (MAP) solution of a probabilistic model. This probabilistic interpretation sets the stage for introducing our probabilistic model in the next section.

We first introduce the generalized normal distribution [11] which is useful for the model to be introduced.

**Definition 1** *$z$ is a univariate generalized normal random variable iff its probability density function (p.d.f.) is given as follows:*

$$p(z) = \frac{1}{2\rho\Gamma(1 + \frac{1}{q})} \exp\Big( -\frac{|z - \mu|^q}{\rho^q} \Big),$$

*where $\Gamma(\cdot)$ denotes the Gamma function and $|\cdot|$ denotes the absolute value of a scalar.*

For simplicity, if $z$ is a univariate generalized normal random variable, we write $z \sim \mathcal{GN}(\mu, \rho, q)$. The (ordinary) normal distribution can be viewed as a special case of the generalized normal distribution when $q = 2$ and the Laplace distribution is a special case when $q = 1$. When $q$ approaches $+\infty$, the generalized normal distribution approaches the uniform distribution in the range $[\mu - \rho, \mu + \rho]$. The generalized normal distribution has proven useful in Bayesian analysis and robustness studies.

**Definition 2** *A standardized $r \times 1$ multivariate generalized normal random variable $\mathbf{z} = (z_1, \ldots, z_r)^T$ consists of $r$ independent and identically distributed (i.i.d.) univariate generalized normal random variables.*

If $\mathbf{z}$ is a standardized $r \times 1$ multivariate generalized normal random variable, we write $\mathbf{z} \sim \mathcal{MGN}(\mu, \rho, q)$ with the following p.d.f.:

$$p(\mathbf{z}) = \frac{1}{\big[2\rho\Gamma(1 + \frac{1}{q})\big]^r} \exp\Big( -\frac{\sum_{i=1}^{r} |z_i - \mu|^q}{\rho^q} \Big).$$

With these definitions, we now begin to present our probabilistic interpretation for multi-task feature selection by proposing a probabilistic model. For notational simplicity, we assume that all tasks perform regression. Extension to include classification tasks will go through similar derivation.

For a regression problem, we use the normal distribution to define the likelihood for $\mathbf{x}_j^i$:

$$y_j^i \sim \mathcal{N}(\mathbf{w}_i^T \mathbf{x}_j^i + b_i, \sigma^2), \tag{2}$$

where $\mathcal{N}(\mu, s^2)$ denotes the (univariate) normal distribution with mean $\mu$ and variance $s^2$.

We impose the generalized normal prior on each element of $\mathbf{W}$:

$$w_{ij} \sim \mathcal{GN}(0, \rho_i, q), \tag{3}$$

where $w_{ij}$ is the $(i, j)$th element of $\mathbf{W}$ (or, equivalently, the $i$th element of $\mathbf{w}_j$ or the $j$th element of $\mathbf{w}^i$). Then we can express the prior on $\mathbf{w}^i$ as

$$(\mathbf{w}^i)^T \sim \mathcal{MGN}(0, \rho_i, q).$$

When $q = 2$, this becomes the ARD prior [9, 3, 26] commonly used in Bayesian methods for enforcing sparsity. From this view, the generalized normal prior can be viewed as a generalization of the ARD prior.

With the above likelihood and prior, we can obtain the MAP solution of $\mathbf{W}$ by solving the following problem:

$$\min_{\mathbf{W}, \mathbf{b}, \boldsymbol{\rho}} \; J = \frac{1}{\sigma^2} \sum_{i=1}^{m} \sum_{j=1}^{n_i} L(y_j^i, \mathbf{w}_i^T \mathbf{x}_j^i + b_i) + \sum_{i=1}^{d} \Big( \frac{\|\mathbf{w}^i\|_q^q}{\rho_i^q} + m \ln \rho_i \Big), \tag{4}$$

where $\mathbf{b} = (b_1, \ldots, b_m)^T$ and $\boldsymbol{\rho} = (\rho_1, \ldots, \rho_m)^T$.

We set the derivative of $J$ with respect to $\rho_i$ to zero and get

$$\rho_i = \Big(\frac{q}{m}\Big)^{1/q} \|\mathbf{w}^i\|_q.$$

Plugging this into problem (4), the optimization problem can be reformulated as

$$\min_{\mathbf{W}, \mathbf{b}} \; J = \frac{1}{\sigma^2} \sum_{i=1}^{m} \sum_{j=1}^{n_i} L(y_j^i, \mathbf{w}_i^T \mathbf{x}_j^i + b_i) + m \sum_{i=1}^{d} \ln \|\mathbf{w}^i\|_q. \tag{5}$$

Note that problem (5) is non-convex since the second term is non-convex with respect to $\mathbf{W}$. Because $\ln z \leq z - 1$ for any $z > 0$, problem (5) can be relaxed to problem (1) by setting $\lambda = m\sigma^2$.

So the solutions of multi-task feature selection methods can be viewed as the solution of the relaxed optimization problem above. In many previous works such as [5, 27], $\ln(x)$ can be used as an approximation of $I(x \neq 0)$ where $I(\cdot)$ is an indicator function. Using this view, we can regard the second term in problem (5) as an approximation of the number of rows with nonzero $q$-norms.

Note that we can directly solve problem (5) using a majorization-minimization (MM) algorithm [14]. For numerical stability, we can slightly modify the objective function in problem (5) by replacing the second term with $m \sum_{i=1}^{d} \ln(\|\mathbf{w}^i\|_q + \alpha)$ where $\alpha$ can be regarded as a regularization parameter. We denote the solution obtained in the $k$th iteration as $\mathbf{w}^i_{(k)}$. In the $(k+1)$th iteration, due to the concavity property of $\ln(\cdot)$, we can bound the second term in problem (5) as follows

$$\sum_{i=1}^{d} \ln(\|\mathbf{w}^i\|_q + \alpha) \leq \sum_{i=1}^{d} \left[ \ln(\|\mathbf{w}^i_{(k)}\|_q + \alpha) + \frac{\|\mathbf{w}^i\|_q - \|\mathbf{w}^i_{(k)}\|_q}{\|\mathbf{w}^i_{(k)}\|_q + \alpha} \right].$$

Thus, in the $(k+1)$th iteration, we need to solve a weighted version of problem (1):

$$\min_{\mathbf{W}, \mathbf{b}} \frac{1}{\sigma^2} \sum_{i=1}^{m} \sum_{j=1}^{n_i} L(y_j^i, \mathbf{w}_i^T \mathbf{x}_j^i + b_i) + m \sum_{i=1}^{d} \frac{\|\mathbf{w}^i\|_q}{\|\mathbf{w}^i_{(k)}\|_q + \alpha}.$$

According to [14], the MM algorithm is guaranteed to converge to a local optimum.

# 4 A Probabilistic Framework for Multi-Task Feature Selection

In the probabilistic interpretation above, we use a type II method to estimate $\{\rho_i\}$ in the generalized normal prior which can be viewed as a generalization of the ARD prior. In the ARD prior, according to [19], this approach is likely to lead to overfitting because the hyperparameters in the ARD prior are treated as points. Similar to the ARD prior, the model in the above section may overfit since $\{\rho_i\}$ are estimated via point estimation. In the following, we will present our probabilistic framework for multi-task feature selection by imposing priors on the hyperparameters.

## 4.1 The Model

As in the above section, the likelihood for $\mathbf{x}_j^i$ is also defined based on the normal distribution:

$$y_j^i \sim \mathcal{N}(\mathbf{w}_i^T \mathbf{x}_j^i + b_i, \sigma_i^2). \tag{6}$$

Here we use different noise variances $\sigma_i$ for different tasks to make our model more flexible. The prior on $\mathbf{W}$ is also defined similarly:

$$w_{ij} \sim \mathcal{GN}(0, \rho_i, q). \tag{7}$$

The main difference here is that we treat $\rho_i$ as a random variable with the noninformative Jeffreys prior:

$$p(\rho_i) \propto \sqrt{I(\rho_i)} = \sqrt{\mathbb{E}_{\mathbf{w}^i | \rho_i} \left[ \left( \frac{\partial \ln p(\mathbf{w}^i | \rho_i)}{\partial \rho_i} \right)^2 \right]} \propto \frac{1}{\rho_i}, \tag{8}$$

where $I(\rho_i)$ denotes the Fisher information for $\rho_i$ and $\mathbb{E}_\theta[\cdot]$ denotes the expectation with respect to $\theta$. One advantage of using the Jeffreys prior is that the distribution has no hyperparameters.

## 4.2 Parameter Learning and Inference

Here we use the EM algorithm [8] to learn the model parameters. In our model, we denote $\boldsymbol{\Theta} = \{\mathbf{W}, \mathbf{b}, \{\sigma_i\}, q\}$ as the model parameters and $\boldsymbol{\rho} = (\rho_1, \ldots, \rho_d)^T$ as the hidden variables.

In the E-step, we construct the so-called $Q$-function as the surrogate for the log-likelihood:

$$Q(\boldsymbol{\Theta} | \boldsymbol{\Theta}^{(k)}) = \int \ln p(\boldsymbol{\Theta} | \mathbf{y}, \boldsymbol{\rho}) p(\boldsymbol{\rho} | \mathbf{y}, \boldsymbol{\Theta}^{(k)}) d\boldsymbol{\rho},$$

where $\boldsymbol{\Theta}^{(k)}$ denotes the estimate of $\boldsymbol{\Theta}$ in the $k$th iteration and $\mathbf{y} = (y_1^1, \ldots, y_{n_m}^m)^T$. It is easy to show that

$$\ln p(\boldsymbol{\Theta} | \mathbf{y}, \boldsymbol{\rho}) \propto \ln p(\mathbf{y} | \mathbf{W}, \{\sigma_i\}) + \ln p(\mathbf{W} | \boldsymbol{\rho})$$

$$\propto - \sum_{i=1}^{m} \left[ \sum_{j=1}^{n_i} \frac{(y_j^i - \mathbf{w}_i^T \mathbf{x}_j^i - b_i)^2}{2\sigma_i^2} + \frac{n_i \ln \sigma_i^2}{2} \right] - \sum_{i=1}^{d} \frac{1}{\rho_i^q} \sum_{j=1}^{m} |w_{ij}|^q - md \ln \Gamma(1 + \frac{1}{q})$$

and $p(\boldsymbol{\rho}|\mathbf{y}, \boldsymbol{\Theta}^{(k)}) \propto \prod_{i=1}^{d} \left( p(\rho_i) p(\mathbf{w}_{(k)}^i|\rho_i) \right)$. We then compute $\mathbb{E}[\frac{1}{\rho_i^q}|\mathbf{y}, \boldsymbol{\Theta}^{(k)}]$ as

$$\mathbb{E}\left[\frac{1}{\rho_i^q}\Big|\mathbf{y}, \boldsymbol{\Theta}^{(k)}\right] = \frac{\int_0^\infty \frac{1}{\rho_i^q} p(\rho_i) p(\mathbf{w}_{(k)}^i|\rho_i) d\rho_i}{\int_0^\infty p(\rho_i) p(\mathbf{w}_{(k)}^i|\rho_i) d\rho_i} = \frac{m}{q\|\mathbf{w}_{(k)}^i\|_q^q}.$$

So we can get

$$Q(\boldsymbol{\Theta}|\boldsymbol{\Theta}^{(k)}) = -\sum_{i=1}^{m}\left[\sum_{j=1}^{n_i} \frac{(y_j^i - \mathbf{w}_i^T\mathbf{x}_j^i - b_i)^2}{2\sigma_i^2} + \frac{n_i \ln \sigma_i^2}{2}\right] - \sum_{i=1}^{d}\beta_i\sum_{j=1}^{m}|w_{ij}|^q - md\ln\Gamma(1+\frac{1}{q}),$$

where $\beta_i = \frac{m}{q\|\mathbf{w}_{(k)}^i\|_q^q}$.

In the M-step, we maximize $Q(\boldsymbol{\Theta}|\boldsymbol{\Theta}^{(k)})$ to update the estimates of $\mathbf{W}$, $\mathbf{b}$, $\{\sigma_i\}$ and $q$.

For the estimation of $\mathbf{W}$, we need to solve $m$ convex optimization problems

$$\min_{\mathbf{w}_i} \ J = \beta_0\|\hat{\mathbf{y}}_i - \mathbf{X}_i^T\mathbf{w}_i\|_2^2 + \sum_{j=1}^{d}\beta_j|w_{ji}|^q, \qquad i = 1, \ldots, m, \tag{9}$$

where $\hat{\mathbf{y}}_i = (y_1^i - b_i^{(k)}, \ldots, y_{n_i}^i - b_i^{(k)})^T$, $\mathbf{X}_i = (\mathbf{x}_1^i, \ldots, \mathbf{x}_{n_i}^i)$, and $\beta_0 = \frac{1}{2(\sigma_i^{(k)})^2}$. When $q = 2$, this becomes the conventional ridge regression problem. Here $\beta_j$ is related to the sparsity of the $j$th row in $\mathbf{W}^{(k)}$: the more sparse the $j$th row in $\mathbf{W}^{(k)}$, the larger the $\beta_j$. When $\beta_j$ is large, $w_{ji}$ will be enforced to approach 0. We use a gradient method such as conjugate gradient to optimize problem (9). The subgradient with respect to $\mathbf{w}_i$ is

$$\frac{\partial J}{\partial \mathbf{w}_i} = 2\beta_0\left(\mathbf{X}_i\mathbf{X}_i^T\mathbf{w}_i - \mathbf{X}_i\hat{\mathbf{y}}_i\right) + q\boldsymbol{\theta},$$

where $\boldsymbol{\theta} = (\beta_1|w_{1i}|^{q-1}\mathrm{sign}(w_{1i}), \ldots, \beta_d|w_{di}|^{q-1}\mathrm{sign}(w_{di}))^T$ and $\mathrm{sign}(\cdot)$ denotes the sign function.

We set the derivatives of $Q(\boldsymbol{\Theta}|\boldsymbol{\Theta}^{(k)})$ with respect to $\sigma_i$ and $b_i$ to 0 and get

$$b_i^{(k+1)} = \frac{1}{n_i}\sum_{j=1}^{n_i}\left[y_j^i - (\mathbf{w}_i^{(k+1)})^T\mathbf{x}_j^i\right]$$

$$\sigma_i^{(k+1)} = \sqrt{\frac{1}{n_i}\sum_{j=1}^{n_i}\left[y_j^i - (\mathbf{w}_i^{(k+1)})^T\mathbf{x}_j^i - b_i^{(k+1)}\right]^2}.$$

For the estimation of $q$, we also use a gradient method. The gradient can be calculated as

$$\frac{\partial Q}{\partial q} = -\sum_{i=1}^{d}\beta_j\sum_{j=1, w_{ij}^{(k+1)}\neq 0}^{m}|w_{ij}^{(k+1)}|^q\ln|w_{ij}^{(k+1)}| + \frac{md}{q} + \frac{md}{q^2}\psi(\frac{1}{q}),$$

where $\psi(x) \equiv \frac{\partial\ln\Gamma(x)}{\partial x}$ is the digamma function.

## 4.3 Extension to Deal with Outlier Tasks and Tasks with Negative Correlation

An underlying assumption of multi-task feature selection using the $l_{1,q}$ norm is that all tasks are similar to each other and they share the same features. This assumption may not be correct in practice because there may exist outlier tasks (i.e., tasks that are not related to all other tasks) or tasks with negative correlation (i.e., tasks that are negatively correlated with some other tasks). In this section, we will discuss how to extend our probabilistic model to deal with these tasks.

We first introduce the matrix variate generalized normal distribution [13] which is a generalization of the generalized normal distribution to random matrices.

**Definition 3** *A matrix* $\mathbf{Z} \in \mathbb{R}^{s\times t}$ *is a matrix variate generalized normal random variable iff its p.d.f. is given as follows:*

$$p(\mathbf{Z}|\mathbf{M}, \boldsymbol{\Sigma}, \boldsymbol{\Omega}, q) = \frac{1}{\left[2\Gamma(1+\frac{1}{q})\right]^{st}det(\boldsymbol{\Sigma})^t det(\boldsymbol{\Omega})^s}\exp\left[-\sum_{i=1}^{s}\sum_{j=1}^{t}\left|\sum_{k=1}^{s}\sum_{l=1}^{t}(\Sigma^{-1})_{ik}(Z_{kl} - M_{kl})(\Omega^{-1})_{lj}\right|^q\right],$$

*where* $\boldsymbol{\Sigma} \in \mathbb{R}^{s\times s}$ *and* $\boldsymbol{\Omega} \in \mathbb{R}^{t\times t}$ *are nonsingular,* $det(\cdot)$ *denotes the determinant of a square matrix,* $A_{ij}$ *is the* $(i,j)$*th element of matrix* $\mathbf{A}$ *and* $(A^{-1})_{ij}$ *is the* $(i,j)$*th element of the matrix inverse* $\mathbf{A}^{-1}$.

We write $\mathbf{Z} \sim \mathcal{MVGN}(\mathbf{M}, \boldsymbol{\Sigma}, \boldsymbol{\Omega}, q)$ for a matrix variate generalized normal random variable $\mathbf{Z}$. When $q = 2$, the matrix variate generalized normal distribution becomes the (ordinary) matrix variate normal distribution [12] with row covariance matrix $\boldsymbol{\Sigma\Sigma}^T$ and column covariance matrix $\boldsymbol{\Omega\Omega}^T$, which has been used before in multi-task learning [4, 32, 31]. From this view, $\boldsymbol{\Sigma}$ is used to model the relationships between the rows of $\mathbf{Z}$ and $\boldsymbol{\Omega}$ is to model the relationships between the columns.

We note that the prior on $\mathbf{W}$ in Eq. (7) can be written as

$$\mathbf{W} \sim \mathcal{MVGN}(\mathbf{0}, \mathrm{diag}((\rho_1, \ldots, \rho_d)^T), \mathbf{I}_m, q),$$

where $\mathbf{0}$ denotes a zero vector or matrix of proper size, $\mathbf{I}_m$ denotes the $m \times m$ identity matrix and $\mathrm{diag}(\cdot)$ converts a vector into a diagonal matrix. In this formulation, it can be seen that the columns of $\mathbf{W}$ (and hence the tasks) are independent of each other. However, the tasks are in general not independent. So we propose to use a new prior on $\mathbf{W}$:

$$\mathbf{W} \sim \mathcal{MVGN}(\mathbf{0}, \mathrm{diag}((\rho_1, \ldots, \rho_d)^T), \boldsymbol{\Omega}, q), \tag{10}$$

where $\boldsymbol{\Omega}$ models the pairwise relationships between tasks.

The likelihood is still based on the normal distribution. Since in practice the relationships between tasks are not known in advance, we also need to estimate $\boldsymbol{\Omega}$ from data.

For parameter learning, we again use the EM algorithm to learn the model parameters. Here the model parameters are denoted as $\boldsymbol{\Theta} = \{\mathbf{W}, \mathbf{b}, \{\sigma_i\}, q, \boldsymbol{\Omega}\}$. It is easy to show that

$$\ln p(\boldsymbol{\Theta}|\mathbf{y}, \boldsymbol{\rho}) \propto -\sum_{i=1}^{m}\Big[\sum_{j=1}^{n_i}\frac{(y_j^i - \mathbf{w}_i^T\mathbf{x}_j^i - b_i)^2}{2\sigma_i^2} + \frac{n_i\ln\sigma_i^2}{2}\Big] - \sum_{i=1}^{d}\frac{1}{\rho_i^q}\sum_{j=1}^{m}\Big|\sum_{l=1}^{m}W_{il}(\Omega^{-1})_{lj}\Big|^q$$
$$- md\ln\Gamma(1 + \frac{1}{q}) - d\ln\det(\boldsymbol{\Omega}).$$

Then we compute $\mathbb{E}[\frac{1}{\rho_i^q}|\mathbf{y}, \boldsymbol{\Theta}^{(k)}]$ as

$$\mathbb{E}\Big[\frac{1}{\rho_i^q}\Big|\mathbf{y}, \boldsymbol{\Theta}^{(k)}\Big] = \frac{\int_0^\infty \frac{1}{\rho_i^q}p(\rho_i)p(\mathbf{w}_{(k)}^i|\rho_i)d\rho_i}{\int_0^\infty p(\rho_i)p(\mathbf{w}_{(k)}^i|\rho_i)d\rho_i} = \frac{m}{q\sum_{j=1}^m\big|\sum_{l=1}^m W_{il}^{(k)}((\Omega^{(k)})^{-1})_{lj}\big|^q} \equiv \alpha_i^{(k)}.$$

In the E-step, the $Q$-function can be formulated as

$$Q(\boldsymbol{\Theta}|\boldsymbol{\Theta}^{(k)}) = -\sum_{i=1}^{m}\Big[\sum_{j=1}^{n_i}\frac{(y_j^i - \mathbf{w}_i^T\mathbf{x}_j^i - b_i)^2}{2\sigma_i^2} + \frac{n_i\ln\sigma_i^2}{2}\Big] - \sum_{i=1}^{d}\alpha_i^{(k)}\sum_{j=1}^{m}\Big|\sum_{l=1}^{m}W_{il}(\Omega^{-1})_{lj}\Big|^q$$
$$- md\ln\Gamma(1 + \frac{1}{q}) - d\ln\det(\boldsymbol{\Omega}).$$

In the M-step, for $\mathbf{W}$ and $\boldsymbol{\Omega}$, the optimization problem becomes

$$\min_{\mathbf{W},\boldsymbol{\Omega}} \sum_{i=1}^{m}\gamma_i^{(k)}\sum_{j=1}^{n_i}(\hat{y}_j^i - \mathbf{w}_i^T\mathbf{x}_j^i)^2 + \sum_{i=1}^{d}\alpha_i^{(k)}\sum_{j=1}^{m}\Big|\sum_{l=1}^{m}W_{il}(\Omega^{-1})_{lj}\Big|^q + d\ln\det(\boldsymbol{\Omega}),$$

where $\gamma_i^{(k)} = \frac{1}{2(\sigma_i^{(k)})^2}$. We define a new variable $\hat{\mathbf{W}} = \mathbf{W}\boldsymbol{\Omega}^{-1}$ to rewrite the above problem as

$$\min_{\hat{\mathbf{W}},\boldsymbol{\Omega}} F = \sum_{i=1}^{m}\gamma_i^{(k)}\sum_{j=1}^{n_i}(\hat{y}_j^i - \mathbf{e}_i^T\boldsymbol{\Omega}^T\hat{\mathbf{W}}^T\mathbf{x}_j^i)^2 + \sum_{i=1}^{d}\alpha_i^{(k)}\sum_{j=1}^{m}|\hat{w}_{ij}|^q + d\ln\det(\boldsymbol{\Omega}),$$

where $\mathbf{e}_i$ denotes the $i$th column of the $m \times m$ identity matrix. We use an alternating method to solve this problem. For a fixed $\boldsymbol{\Omega}$, the problem with respect to $\hat{\mathbf{W}}$ is a convex problem and we use conjugate gradient to solve it with the following subgradient

$$\frac{\partial F}{\partial \hat{\mathbf{W}}} = 2\sum_{i=1}^{m}\gamma_i^{(k)}\sum_{j=1}^{n_i}\Big[\mathbf{x}_j^i(\mathbf{x}_j^i)^T\hat{\mathbf{W}}\boldsymbol{\Omega}\mathbf{e}_i\mathbf{e}_i^T\boldsymbol{\Omega}^T - y_j^i\mathbf{x}_j^i\mathbf{e}_i^T\boldsymbol{\Omega}^T\Big] + q\mathbf{M},$$

where $\mathbf{M}$ is a $d \times m$ matrix with the $(i, j)$th element $\alpha_i^{(k)}|\hat{w}_{ij}|^{q-1}\mathrm{sign}(\hat{w}_{ij})$. For a fixed $\hat{\mathbf{W}}$, we also use conjugate gradient with the following gradient

$$\frac{\partial F}{\partial \boldsymbol{\Omega}} = 2\sum_{i=1}^{m}\gamma_i^{(k)}\sum_{j=1}^{n_i}\Big[\hat{\mathbf{W}}^T\mathbf{x}_j^i(\mathbf{x}_j^i)^T\hat{\mathbf{W}}\boldsymbol{\Omega}\mathbf{e}_i\mathbf{e}_i^T - y_j^i\hat{\mathbf{W}}^T\mathbf{x}_j^i\mathbf{e}_i^T\Big] + d(\boldsymbol{\Omega}^T)^{-1}.$$

After obtaining the optimal $\hat{\mathbf{W}}^\star$ and $\boldsymbol{\Omega}^\star$, we can compute the optimal $\mathbf{W}^\star$ as $\mathbf{W}^\star = \hat{\mathbf{W}}^\star\boldsymbol{\Omega}^\star$. The update rules for $\{\sigma_i\}$, $\{b_i\}$ and $q$ are similar to those in the above section.

# 5  Related Work

Some probabilistic multi-task feature selection methods have been proposed before [28, 2]. However, they only focus on the $l_{1,2}$ norm. Moreover, they use point estimation in the ARD prior and hence, as discussed in Section 3, are susceptible to overfitting [19].

Zhang et al. [30] proposed a latent variable model for multi-task learning by using the Laplace prior to enforce sparsity. This is equivalent to using the $L_{1,1}$ norm in our framework which, as discussed above, cannot enforce group sparsity among different features over all tasks.

# 6  Experiments

In this section, we study our methods empirically on two cancer classification applications using microarray gene expression data. We compare our methods with three related methods: multi-task feature learning (MTFL) [1][1], multi-task feature selection using $l_{1,2}$ regularization [16][2], and multi-task feature selection using $l_{1,\infty}$ regularization [20][3].

## 6.1  Breast Cancer Classification

We first conduct empirical study on a breast cancer classification application. This application consists of three learning tasks with data collected under different platforms [21]. The dataset for the first task, collected at the Koo Foundation Sun Yat-Sen Cancer Centre in Taipei, contains 89 samples with 8948 genes per sample. The dataset for the second task, obtained from the Netherlands Cancer Institute, contains 97 samples with 16360 genes per sample. Most of the patients in this dataset had stage I or II breast cancer. The dataset for the third task, obtained using 22K Agilent oligonucleotide arrays, contains 114 samples with 12065 genes per sample. Even though these three datasets were collected under different platforms, they share 6092 common genes which are used in our experiments.

Here we abbreviate the method in Section 4.2 as PMTFS1 and that in Section 4.3 as PMTFS2. For each task, we choose 70% of the data for training and the rest for testing. We perform 10 random splits of the data and report the mean and standard derivation of the classification error over the 10 trials. The results are summarized in Table 1. It is clear that PMTFS1 outperforms the three previous methods, showing the effectiveness of our more general formulation with $q$ determined automatically. Moreover, we also note that PMTFS2 is better than PMTFS1. This verifies the usefulness of exploiting the relationships between tasks in multi-task feature selection. Since our methods can estimate $q$ automatically, we compute the mean of the estimated $q$ values over 10 trials. The means for PMTFS1 and PMTFS2 are 2.5003 and 2.6718, respectively, which seem to imply that smaller values of $q$ are preferred for this application. This probably explains why the performance of $\text{MTFS}_{1,\infty}$ is not good when compared with other methods.

Table 1: Comparison of different methods on the breast cancer classification application in terms of classification error rate (in mean±std-dev). Each column in the table represents one task.

| Method | 1st Task | 2nd Task | 3rd Task |
|---|---|---|---|
| MTFL | 0.3478±0.1108 | 0.0364±0.0345 | 0.3091±0.0498 |
| $\text{MTFS}_{1,2}$ | 0.3370±0.0228 | 0.0343±0.0134 | 0.2855±0.0337 |
| $\text{MTFS}_{1,\infty}$ | 0.3896±0.0583 | 0.1136±0.0579 | 0.2909±0.0761 |
| PMTFS1 | 0.3072±0.0234 | 0.0298±0.0121 | 0.1786±0.0245 |
| PMTFS2 | 0.2870±0.0228 | 0.0273±0.0102 | 0.1455±0.0263 |

## 6.2  Prostate Cancer Classification

We next study a prostate cancer classification application consisting of two tasks. The Singh dataset [22] for the first task is made up of laser intensity images from each microarray. The RMA preprocessing method was used to produce gene expression values from these images. On the other

hand, the Welsh dataset [25] for the second task is already in the form of gene expression values. Even though the collection techniques for the two datasets are different, they have 12600 genes in common and are used in our experiments.

The experimental setup for this application is similar to that in the previous subsection, that is, 70% of the data of each task are used for training and the rest for testing, and 10 random splits of the data are performed. We report the mean and standard derivation of the classification error over the 10 trials in Table 2. As in the first set of experiments, PMTFS1 and PMTFS2 are better than the other three methods compared and PMTFS2 slightly outperforms PMTFS1. The means of the estimated $q$ values for PMTFS1 and PMTFS2 are 2.5865 and 2.6319, respectively. So it seems that smaller values are also preferred for this application.

Table 2: Comparison of different methods on the prostate cancer classification application in terms of classification error rate (in mean±std-dev). Each column in the table represents one task.

| Method | 1st Task | 2nd Task |
|---|---|---|
| MTFL | 0.1226±0.0620 | 0.3500±0.0085 |
| $\text{MTFS}_{1,2}$ | 0.1232±0.0270 | 0.3420±0.0067 |
| $\text{MTFS}_{1,\infty}$ | 0.2216±0.1667 | 0.4200±0.1304 |
| PMTFS1 | 0.1123±0.0170 | 0.3214±0.0053 |
| PMTFS2 | 0.1032±0.0136 | 0.3000±0.0059 |

## 7   Concluding Remarks

In this paper, we have proposed a probabilistic framework for general multi-task feature selection using the $l_{1,q}$ norm ($1 < q \leq \infty$). Our model allows the optimal value of $q$ to be determined from data automatically. Besides considering the case in which all tasks are similar, we have also considered the more general and challenging case in which there also exist outlier tasks or tasks with negative correlation.

Compressed sensing aims at recovering the sparse signal $\mathbf{w}$ from a measurement vector $\mathbf{b} = \mathbf{Aw}$ for a given matrix $\mathbf{A}$. Compressed sensing can be extended to the multiple measurement vector (MMV) model in which the signals are represented as a set of jointly sparse vectors sharing a common set of nonzero elements [7, 6, 23]. Specifically, joint compressed sensing considers the reconstruction of the signal represented by a matrix $\mathbf{W}$, which is given by a dictionary (or measurement matrix) $\mathbf{A}$ and multiple measurement vector $\mathbf{B}$ such that $\mathbf{B} = \mathbf{AW}$. Similar to multi-task feature selection, we can use $\|\mathbf{W}\|_{1,q}$ to enforce the joint sparsity in $\mathbf{W}$. Since there usually exists noise in the data, the optimization problem of MMV can be formulated as: $\min_{\mathbf{W}} \lambda\|\mathbf{W}\|_{1,q} + \|\mathbf{AW} - \mathbf{B}\|_2^2$. This problem is almost identical to problem (1) except that the loss defines the reconstruction error rather than the prediction error. So we can use the probabilistic model presented in Section 4 to develop a probabilistic model for joint compressed sensing. Besides, we are also interested in developing a full Bayesian version of our model to further exploit the advantages of Bayesian modeling.

## Acknowledgment

This research has been supported by General Research Fund 622209 from the Research Grants Council of Hong Kong.

## Footnotes

[1]http://ttic.uchicago.edu/∼argyriou/code/index.html

[2]http://www.public.asu.edu/∼jye02/Software/SLEP/index.htm

[3]http://www.lsi.upc.edu/∼aquattoni/

## References

[1] A. Argyriou, T. Evgeniou, and M. Pontil. Convex multi-task feature learning. *Machine Learning*, 73(3):243–272, 2008.

[2] J. Bi, T. Xiong, S. Yu, M. Dundar, and R. B. Rao. An improved multi-task learning approach with applications in medical diagnosis. In *ECMLPKDD*, 2008.

[3] C. M. Bishop. *Pattern Recognition and Machine Learning*. Springer, New York, 2006.

[4] E. Bonilla, K. M. A. Chai, and C. Williams. Multi-task Gaussian process prediction. In *NIPS 20*, 2008.

[5] E. J. Candès, M. B. Wakin, and S. P. Boyd. Enhancing sparsity by reweighted $l_1$ minimization. *Journal of Fourier Analysis and Applications*, 14(5):877–905, 2008.

[6] J. Chen and X. Huo. Theoretical results on sparse representations of multiple-measurement vectors. *IEEE Transactions on Signal Processing*, 54(12):4634–4643, 2006.

[7] S. F. Cotter, B. D. Rao, K. Engan, and K. Kreutz-Delgado. Sparse solutions to linear inverse problems with multiple measurement vectors. *IEEE Transactions on Signal Processing*, 53(7):2477–2488, 2005.

[8] A. P. Dempster, N. M. Laird, and D. B. Rubin. Maximum likelihood from incomplete data via the EM algorithm. *Journal of the Royal Statistic Society, B*, 39(1):1–38, 1977.

[9] M. A. T. Figueiredo. Adaptive sparseness for supervised learning. *IEEE Transactions on Pattern Analysis and Machine Intelligence*, 25(9):1150–1159, 2003.

[10] A. Gelman, J. B. Carlin, H. S. Stern, and D. B. Rubin. *Bayesian Data Analysis*. Chapman & Hall, 2nd edition, 2003.

[11] I. R. Goodman and S. Kotz. Multivariate $\theta$-generalized normal distributions. *Journal of Multivariate Analysis*, 3(2):204–219, 1973.

[12] A. K. Gupta and D. K. Nagar. *Matrix Variate Distributions*. Chapman & Hall, 2000.

[13] A. K. Gupta and T. Varga. Matrix variate $\theta$-generalized normal distribution. *Transactions of The American Mathematical Society*, 347(4):1429–1437, 1995.

[14] K. Lange, D. R. Hunter, and I. Yang. Optimization transfer using surrogate objective functions. *Journal of Computational and Graphical Statistics*, 9(1):1–59, 2000.

[15] H. Liu, M. Palatucci, and J. Zhang. Blockwise coordinate descent procedures for the multi-task lasso, with applications to neural semantic basis discovery. In *ICML*, 2009.

[16] J. Liu, S. Ji, and J. Ye. Multi-task feature learning via efficient $l_{2,1}$-norm minimization. In *UAI*, 2009.

[17] G. Obozinski, B. Taskar, and M. Jordan. Multi-task feature selection. Technical report, Department of Statistics, University of California, Berkeley, June 2006.

[18] G. Obozinski1, B. Taskar, and M. I. Jordan. Joint covariate selection and joint subspace selection for multiple classification problems. *Statistics and Computing*, 20(2):231–252, 2010.

[19] Y. Qi, T. P. Minka, R. W. Picard, and Z. Ghahramani. Predictive automatic relevance determination by expectation propagation. In *ICML*, 2004.

[20] A. Quattoni, X. Carreras, M. Collins, and T. Darrell. An efficient projection for $l_{1,\infty}$ regularization. In *ICML*, 2009.

[21] A. A. Shabalin, H. Tjelmeland, C. Fan, C. M. Perou, and A. B. Nobel. Merging two gene-expression studies via cross-platform normalization. *Bioinformatics*, 24(9):1154–1160, 2008.

[22] D. Singh, P. G. Febbo, K. Ross, D. G. Jackson, J. Manola, C. Ladd, P. Tamayo, A. A. Renshaw, A. V. DAmico, J. P. Richie, E. S. Lander, M. Loda, P. W. Kantoff, T. R. Golub, and W. R.Sellers. Gene expression correlates of clinical prostate cancer behavior. *Cancer Cell*, 1(2):203–209, 2002.

[23] L. Sun, J. Liu, J. Chen, and J. Ye. Efficient recovery of jointly sparse vectors. In *NIPS 22*. 2009.

[24] B. A. Turlach, W. N. Wenables, and S. J. Wright. Simultaneous variable selection. *Technometrics*, 47(3):349–363, 2005.

[25] J. B. Welsh, L. M. Sapinoso, A. I. Su, S. G. Kern, J. Wang-Rodriguez, C. A. Moskaluk, F. H. Frierson, Jr., and G. M. Hampton. Analysis of gene expression identifies candidate markers and pharmacological targets in prostate cancer. *Cancer Research*, 61(16):5974–5978, 2001.

[26] D. Wipf and S. Nagarajan. A new view of automatic relevance determination. In *NIPS 20*, 2007.

[27] D.P. Wipf and S. Nagarajan. Iterative reweighted $l_1$ and $l_2$ methods for finding sparse solutions. *Journal of Selected Topics in Signal Processing*, 2010.

[28] T. Xiong, J. Bi, B. Rao, and V. Cherkassky. Probabilistic joint feature selection for multi-task learning. In *SDM*, 2007.

[29] M. Yuan and Y. Lin. Model selection and estimation in regression with grouped variables. *Journal of the Royal Statistical Society, Series B*, 2006.

[30] J. Zhang, Z. Ghahramani, and Y. Yang. Flexible latent variable models for multi-task learning. *Machine Learning*, 73(3):221–242, 2008.

[31] Y. Zhang and D.-Y. Yeung. A convex formulation for learning task relationships in multi-task learning. In *UAI*, 2010.

[32] Y. Zhang and D.-Y. Yeung. Multi-task learning using generalized $t$ process. In *AISTATS*, 2010.

